# Weighted Likelihood Policy Search
# with Model Selection

**Tsuyoshi Ueno** [*]
Japan Science and Technology Agency
ueno@ar.sanken.osaka-u.ac.jp

**Kohei Hayashi**
University of Tokyo
hayashi.kohei@gmail.com

**Takashi Washio**
Osaka University
washio@ar.sanken.osaka-u.ac.jp

**Yoshinobu Kawahara**
Osaka University
kawahara@ar.sanken.osaka-u.ac.jp

## Abstract

Reinforcement learning (RL) methods based on direct policy search (DPS) have been actively discussed to achieve an efficient approach to complicated Markov decision processes (MDPs). Although they have brought much progress in practical applications of RL, there still remains an unsolved problem in DPS related to model selection for the policy. In this paper, we propose a novel DPS method, *weighted likelihood policy search (WLPS)*, where a policy is efficiently learned through the weighted likelihood estimation. WLPS naturally connects DPS to the statistical inference problem and thus various sophisticated techniques in statistics can be applied to DPS problems directly. Hence, by following the idea of the *information criterion*, we develop a new measurement for model comparison in DPS based on the weighted log-likelihood.

## 1 Introduction

In the last decade, several direct policy search (DPS) methods have been developed in the field of reinforcement learning (RL) [1, 2, 3, 4, 5, 6, 7, 8, 9] and have been successfully applied to practical decision making applications [5, 7, 9]. Unlike classical approaches [10], DPS characterizes a policy as a parametric model and explores parameters such that the expected reward is maximized in a given model space. Hence, if one employs a model with a reasonable number of DoF (degrees of freedom), DPS could find a reasonable policy efficiently even when the target decision making task has a huge number of DoF. Therefore, the development of an efficient model selection methodology for the policy is crucial in RL research.

In this paper, we propose *weighted likelihood policy search (WLPS)*: an efficient iterative policy search algorithm that allows us to select an appropriate model automatically from a set of candidate models. To this end, we first introduce a log-likelihood function weighted by the discounted sum of future rewards as the cost function for DPS. In WLPS, the policy parameters are updated by iteratively maximizing the weighted log-likelihood for the obtained sample sequence. A key property of WLPS is that the maximization of weighted log-likelihood corresponds to that of the lower bound of the expected reward and thus, WLPS is guaranteed to increase the expected reward monotonically at each iteration. This can be shown to converge to the same solution as the expectation-maximization policy search (EMPS) [1, 4, 9]. In this way, our framework gives a natural connection between DPS and the statistical inference problem for maximum likelihood estimation. One benefit of this approach is that we can directly apply the *information criterion* scheme [11, 12], which is a familiar theory in statistics, to the weighted log-likelihood. This enables us to construct a model selection strategy for the policy by comparing the information criterion based on the weighted log-likelihood.

The contribution of this paper is summarized as follows:

---

[*]https://sites.google.com/site/tsuyoshiueno/

1. We prove that each update to the policy parameters resulting from the maximization of the weighted log-likelihood has consistency and asymptotic normality, which have been not elucidated yet in DPS, and converges to the same solution as EMPS.

2. We introduce prior distribution on the policy parameter, and analyze the asymptotic behavior of the *marginal weighted likelihood* given by marginalizing out the policy parameter. We then propose a measure of the goodness of the policy model based on the posterior probability of the model in a similar way as *Bayesian information criterion* [12].

The rest of the paper is organized as follows. We first give a formulation of the DPS problem in RL, and a short overview of EMPS (Section 2). Next, we present our new DPS framework, WLPS, and investigate the theoretical aspects thereof (Section 3). In addition, we construct the model selection strategy for the policy (Section 4). Finally, we present a statistical interpretation of WLPS and discuss future directions of study in this regard (Section 5).

**Related Works** Several approaches have been proposed for the model selection problem in the estimation of a state-action value function [13, 14]. [14] derived the PAC-Bayesian bounds for estimating state-action value functions. [13] developed a complexity regularization based model selection algorithm from the viewpoint of the minimization of the Bellman error, and investigated its theoretical aspects. Although these studies allow us to select a good model for a state-value function with theoretical supports, they cannot be applied to model selection for DPS directly. [15] developed a model selection strategy for DPS by reusing the past observed sample sequences through the importance weighted cross-validation (IWCV). However, IWCV requires heavy computational costs and includes computational instability when estimating the importance sampling weights.

Recently, there are a number of studies that reformulate stochastic optimal control and RL as a minimization problem of Kullback-Leiblar (KL) divergence [16, 17, 18]. Our approach is closely related to these works; in fact, WLPS can also be interpreted as the minimization problem of the reverse form of KL divergence compared with that used in [16, 17, 18].

## 2 Preliminary: EMPS

We consider discrete-time infinite horizon Markov Decision Processes (MDPs), defined as the quadruple $(\mathcal{X}, \mathcal{U}, p, r)$: $\mathcal{X} \subseteq \mathbb{R}^{d_x}$ is a state space; $\mathcal{U} \subseteq \mathbb{R}^{d_u}$ is an action space; $p(x'|x, u)$ is a stationary transition distribution to the next state $x'$ when taking action $u$ at state $x$; and $r : \mathcal{X} \times \mathcal{U} \mapsto \mathbb{R}^+$ is a positive reward received with the state transition. Let $\pi_\theta(u|x) := p(u|x, \theta)$ be the stochastic parametrized policy followed by the agent, where an $m$-dimensional vector $\theta \in \Theta, \Theta \subseteq \mathbb{R}^m$ means an adjustable parameter. Given initial state $x_1$ and parameter vector $\theta$, the joint distribution of the sample sequence, $\{x_{2:n}, u_{1:n}\}$, of the MDP is described as

$$p_\theta(x_{2:n}, u_{1:n}|x_1) = \pi_\theta(u_1|x_1) \prod_{i=2}^{n} p(x_i|x_{i-1}, u_{i-1}) \pi_\theta(u_i|x_i). \tag{1}$$

We further impose the following assumptions on MDPs.

**Assumption 1.** *For any $\theta \in \Theta$, the MDP given by Eq. (1), is aperiodic and Harris recurrent. Hence, MDP (1) is ergodic and has a unique invariant stationary distribution $\mu_\theta(x)$, for any $\theta \in \Theta$ [19].*

**Assumption 2.** *For any $x \in \mathcal{X}$ and $u \in \mathcal{U}$, reward $r(x, u)$ is uniformly bounded.*

**Assumption 3.** *Policy $\pi_\theta(u|x)$ is thrice continuously differentiable with respect to parameter $\theta$.*

The general goal of DPS is to find an optimal policy parameter $\theta^* \in \Theta$ that maximizes the expected reward defined by

$$\eta(\theta) := \lim_{n \to \infty} \int\int p_\theta(x_{2:n}, u_{1:n}|x_1) \{R_n\} \, \mathrm{d}x_{2:n} \mathrm{d}u_{1:n}, \tag{2}$$

where $R_n := R_n(x_{1:n}, u_{1:n}) = (1/n) \sum_{i=1}^{n} r(x_i, u_i)$. In general, the direct maximization of objective function (2) forces us to solve a non-convex optimization problem with a high non-linearity. Thus, instead of maximizing Eq. (2), many of the DPS methods maximize the lower bound on Eq. (2), which may be much more tractable than the original objective function.

Lemma 1 shows that there is a tight lower bound on objective function (2).

**Lemma 1.** *[1, 4, 9] The following inequality holds for any distribution $q(x_{2:n}, u_{1:n}|x_1)$:*

$$\ln \eta_n(\theta) \geq \mathcal{F}_n(q, \theta) := \int\int q(x_{2:n}, u_{1:n}|x_1) \left\{ \ln \frac{p_\theta(x_{2:n}, u_{1:n}|x_1) R_n}{q(x_{2:n}, u_{1:n}|x_1)} \right\} \mathrm{d}x_{2:n} \mathrm{d}u_{1:n}, \ \forall n \tag{3}$$

where $\eta_n(\theta) = \iint p(x_{2:n}, u_{1:n}|x_1)\{R_n\}\,\mathrm{d}x_{2:n}\mathrm{d}u_{1:n}$. *The equality holds if $q(x_{2:n}, u_{1:n}|x_1)$ is a maximizer of $\mathcal{F}_n(q, \theta)$ for some $\theta$, i.e., $q^*(x_{2:n}, u_{1:n}|x_1) = \mathrm{argmax}_q \mathcal{F}_n(q, \theta)$, which is satisfied when $q^*(x_{2:n}, u_{1:n}|x_1) \propto p_\theta(x_{2:n}, u_{1:n}|x_1)\{R_n\}$.*

The proof is given in Section 1 in the supporting material. Lemma 1 leads to an effective iterative algorithm, the so-called EMPS, which breaks down the potentially difficult maximization problem for the expected reward into two stages: 1) evaluation of the path distribution $q^*_{\theta'}(x_{2:n}, u_{1:n}|x_1) \propto p_{\theta'}(x_{2:n}, u_{1:n}|x_1)\{R_n\}$ at the current policy parameter $\theta'$, and 2) maximization of $\mathcal{F}_n(q^*_{\theta'}, \theta)$ with respect to parameter $\theta$. This EMPS cycle is guaranteed to increase the value of the expected reward unless the parameters already correspond to a local maximum [1, 4, 9].

Taking the partial derivative of the policy with respect to parameter $\theta$, a new parameter vector $\tilde{\theta}$ that maximizes $\mathcal{F}_n(q^*_{\theta'}, \theta)$ is found by solving the following equation:

$$\iint p_{\theta'}(x_{2:n}, u_{1:n}|x_1) \left( \sum_{i=1}^{n} \psi_{\tilde{\theta}}(x_i, u_i) \right) R_n \mathrm{d}x_{2:n}\mathrm{d}u_{1:n} = 0, \tag{4}$$

where $\psi : \mathcal{X} \times \mathcal{U} \times \Theta$ denotes a partial derivative of the logarithm of the policy with respect to parameter $\theta$, *i.e.*, $\psi_\theta(x, u) := (\partial)/(\partial\theta) \ln \pi_\theta(u|x)$.

Note that if the state transition distribution $p(x'|x, u)$ is known, we can easily derive parameter $\bar{\theta}$ analytically or numerically. However, $p(x'|x, u)$ is generally unknown, and it is a non-trivial problem to identify this distribution in large-scale applications. Thus, it is desirable to find parameter $\bar{\theta}$ in model-free ways, *i.e.*, parameter is estimated from the sample sequences alone, instead of using $p(x'|x, u)$. Although many variants of model-free EMPS algorithms [4, 6, 9, 15] have hitherto been developed, their fundamental theoretical properties such as consistency and asymptotic normality at each iteration have not yet been elucidated. Moreover, it is not even obvious whether they have such desirable statistical properties.

## 3 Proposed framework: WLPS

In this section, we newly introduce a weighted likelihood as the objective function for DPS (Definition 1), and derive the WLPS algorithm, executed by iterating two steps: evaluation and maximization of the weighted log-likelihood function (Algorithm 1). Then, in Section 3.2, we show that WLPS is guaranteed to increase the expected reward at each iteration, and to converge to the same solution as EMPS (Theorem 1).

### 3.1 Overview of WLPS

In this study, we consider the following weighted likelihood function.

**Definition 1.** *Suppose that given initial state $x_1$, a random sequence $\{x_{2:n}, u_{1:n}\}$ is generated from model $p_{\theta'}(x_{2:n}, u_{1:n}|x_1)$ of the MDP. Then, we define a weighted likelihood function $\hat{p}_{\theta',\theta}(x_{2:n}, u_{1:n}|x_1)$ and a weighted log-likelihood function $L_n^{\theta'}(\theta)$, respectively, as*

$$\hat{p}_{\theta',\theta}(x_{2:n}, u_{1:n}|x_1) := \pi_\theta(u_1|x_1)^{Q_1^\beta} \prod_{i=2}^{n} \pi_\theta(u_i|x_i)^{Q_i^\beta} p(x_i|x_{i-1}, u_{i-1}) \tag{5}$$

$$L_n^{\theta'}(\theta) := \ln \hat{p}_{\theta',\theta}(x_{2:n}, u_{1:n}|x_1) := \sum_{i=1}^{n} Q_i^\beta \ln \pi_\theta(u_i|x_i) + \sum_{i=2}^{n} \ln p(x_i|x_{i-1}, u_{i-1}), \tag{6}$$

*where $Q_i^\beta$ is the discounted sum of the future rewards given by $Q_i^\beta := \sum_{j=i}^{n} \beta^{j-i} r(x_j, u_j)$, and $\beta$ is a discounted factor such that $\beta \in [0, 1)$.*

Now, let us consider the maximization of weighted log-likelihood function (6). Taking the partial derivative of weighted log-likelihood (6) with respect to parameter $\theta$, we can obtain the maximum weighted log-likelihood estimator $\hat{\theta}_n := \hat{\theta}(x_{1:n}, u_{1:n})$ as a solution of the following estimation equation:

$$G_n^{\theta'}(\hat{\theta}_n) := \sum_{i=1}^{n} \psi_{\hat{\theta}_n}(x_i, u_i) Q_i^\beta = \sum_{i=1}^{n} \sum_{j=i}^{n} \beta^{j-i} \psi_{\hat{\theta}_n}(x_i, u_i) r(x_j, u_j) = 0. \tag{7}$$

Note that when policy $\pi_\theta$ is modeled by an exponential family, estimating equation (7) can easily be solved analytically or numerically using convex optimization techniques. In WLPS, the update of the policy parameter is performed by evaluating estimating equation (7) and finding estimator $\hat{\theta}_n$ iteratively from this equation. Algorithm 1 gives an outline of the WLPS procedure.

**Algorithm 1** (WLPS)**.**

1. *Generate a sample sequence $\{x_{1:n}, u_{1:n}\}$ by employing the current policy parameter $\theta$, and evaluate estimating equation (7).*

2. *Find a new estimator by solving estimating equation (7) and check for convergence. If convergence is not satisfied, return to step 1.*

It should be noted that WLPS guarantees to monotonically increase the expected reward $\eta(\theta)$, and to converge asymptotically under certain conditions to the same solution as EMPS, given by Eq. (4). In the next subsection, we discuss the reason why WLPS satisfies such desirable statistical properties.

## 3.2 Convergence of WLPS

To begin with, we show consistency and asymptotic normality of estimator $\hat{\theta}_n$ given by Eq. (7) when $\beta$ is any constant between 0 and 1. To this end, we first introduce the notion of *uniform mixing*, which plays an important role when discussing statistical properties in stochastic processes [19]. The definition of uniform mixing is given below.

**Definition 2.** *Let $\{Y_i : i = \{\cdots, -1, 0, 1, \cdots\}\}$ be a strictly stationary process on a probabilistic space $(\Omega, F, P)$, and $F_k^m$ be the $\sigma$-algebra generated by $\{Y_k, \cdots, Y_m\}$. Then, process $\{Y_i\}$ is said to be uniform mixing ($\varphi$-mixing) if $\varphi(s) \to 0$ as $s \to \infty$, where*

$$\varphi(s) := \sup_{A \in \mathcal{F}_{-\infty}^k, B \in \mathcal{F}_{k+s}^\infty} |P(B|A) - P(B)| = 0, \quad P(A) \neq 0.$$

Function $\varphi(s)$ is called the *mixing coefficient*, and if the mixing coefficient converges to zero exponentially fast, *i.e.*, there exist constants $D > 0$ and $\rho \in [0, 1)$ such that $\varphi(s) < D\rho^s$, then the stochastic process is called geometrically uniform mixing. Note that if a stochastic process is a strictly stationary finite-state Markov process and ergodic, the process satisfies the geometrically uniform mixing conditions [19].

Now, we impose certain conditions for proving the consistency and asymptotic normality of estimator $\hat{\theta}_n$, summarized as follows.

**Assumption 4.** *For any $\theta \in \Theta$, MDP $p_\theta(x_{2:n}, u_{1:n}|x_1)$ is geometrically uniform mixing.*

**Assumption 5.** *For any $x \in \mathcal{X}$, $u \in \mathcal{U}$, and $\theta \in \Theta$, function $\psi_\theta(x, u)$ is uniformly bounded.*

**Assumption 6.** *For any $\theta \in \Theta$, there exists a parameter value $\bar{\theta} \in \Theta$ such that*

$$\mathbb{E}_{x_1 \sim \mu_\theta}^{\pi_\theta} \left[ \psi_{\bar{\theta}}(x_1, u_1) \sum_{j=1}^\infty \beta^{j-1} r(x_j, u_j) \right] = 0, \tag{8}$$

*where $\mathbb{E}_{x_1 \sim \mu_\theta}^{\pi_\theta}[\cdot]$ denotes the expectation over $\{x_{2:\infty}, u_{1:\infty}\}$ with respect to distribution $\lim_{n\to\infty} \mu_\theta(x_1)\pi_\theta(u_1|x_1) \prod_{i=2}^n p(x_i|x_i, u_i)\pi_\theta(u_i|x_i)$.*

**Assumption 7.** *For any $\theta \in \Theta$ and $\epsilon > 0$,*

$$\sup_{\theta': |\theta' - \bar{\theta}| > \epsilon} \left| \mathbb{E}_{x_1 \sim \mu_\theta}^{\pi_\theta} \left[ \psi_{\theta'}(x_1, u_1) \sum_{j=1}^\infty \beta^{j-1} r(x_j, u_j) \right] \right| > 0.$$

**Assumption 8.** *For any $\theta \in \Theta$, matrix $\mathrm{A} := \mathrm{A}(\bar{\theta}) = \mathbb{E}_{x_1 \sim \mu_1}^{\pi_\theta} \left[ \mathrm{K}_{\bar{\theta}}(x_1, u_1) \sum_{j=1}^\infty \beta^{j-1} r(x_j, u_j) \right]$ is invertible, where $\mathrm{K}_\theta(x, u) := \partial_\theta \psi_\theta(x, u) = \partial^2 / (\partial\theta\partial\theta^\top) \ln \pi_\theta(u|x)$.*

Under the conditions given in Assumptions 1-7, estimator $\hat{\theta}_n$ converges to $\bar{\theta}$ in probability, as shown in the following lemma.

**Lemma 2.** *Suppose that given initial state $x_1$, a random sequence $\{x_{2:n}, u_{1:n}\}$ is generated from model $\{p_\theta(x_{2:n}, u_{1:n}|x_1)|\theta\}$ of the MDP. If Assumptions 1-7 are satisfied, then estimator $\hat{\theta}_n$ given by estimating equation (7) shows consistency, i.e., estimator $\hat{\theta}_n$ converges to parameter $\bar{\theta}$ in probability.*

The proof is given in Section 2 in the supporting material. Note that if the policy is characterized as an exponential family, we can replace Assumption 7 with Assumption 8 to prove the result in Lemma 3. Next, we show the asymptotic convergence rate of the estimator given a consistent estimator. Lemma 3 shows that the estimator converges at the rate $O_p(n^{-1/2})$.

**Lemma 3.** *Suppose that given initial state $x_1$, a random sequence $\{x_{2:n}, u_{1:n}\}$ is generated from model $p_{\theta'}(x_{2:n}, u_{1:n}|x_1)$, and Assumptions 1-6 and 8 are satisfied. If estimator $\hat{\theta}_n$, given by estimating equation (7) converges to $\bar{\theta}$ in probability, then we have*

$$\sqrt{n}(\hat{\theta}_n - \bar{\theta}) = -\frac{1}{\sqrt{n}} A^{-1} \sum_{i=1}^{n} \sum_{j=i}^{n} \beta^{j-i} \psi_{\bar{\theta}}(x_i, u_i) r(x_j, u_j) + o_p(1). \tag{9}$$

*Furthermore, the right hand side of Eq. (9) converges to a Gaussian distribution whose mean and covariance are, respectively, zero and $A^{-1}\Sigma(A^{-1})^\top$, where $\Sigma := \Sigma(\bar{\theta}) = \Gamma(\bar{\theta}) + \sum_{i=2}^{\infty} \Gamma_i(\bar{\theta}) + \sum_{j=2}^{\infty} \Gamma_j(\bar{\theta})^\top$. Here, $\Gamma_i(\bar{\theta}) := \mathbb{E}_{x_1 \sim \mu_{\theta'}}^{\pi_{\theta'}} \left[ \left( \sum_{j=1}^{\infty} \beta^{j-1} r(x_j, u_j) \right) \left( \sum_{j'=1+i}^{\infty} \beta^{j'-1} r(x_{j'}, u_{j'}) \right) \psi_{\bar{\theta}}(x_1, u_1) \psi_{\bar{\theta}}(x_i, u_i)^\top \right]$.*

The proof is given in Section 3 in the supporting material.

Now we consider the relation between WLPS and EMPS. The following theorem shows that the estimator $\hat{\theta}_n$ given by Eq. (7) converges to the same solution as that of EMPS asymptotically, when taking the limit of $\beta$ to 1.

**Theorem 1.** *Suppose that Assumptions 1-7 are satisfied. If $\beta$ approaches to 1 from below, WLPS leads to the same solution with EMPS given by Eq. (4) as $n \to \infty$[1].*

*Proof.* We introduce the following support lemma.

**Lemma 4.** *Suppose that Assumptions 1-6 are satisfied. Then, the partial derivative of the lower bound with $q_{\theta'}^*$, satisfies*

$$\lim_{n \to \infty} \frac{\partial}{\partial \theta} \mathcal{F}_n(q_{\theta'}^*, \theta) = \lim_{\beta \to 1^-} \mathbb{E}_{x_1 \sim \mu_{\theta'}}^{\pi_{\theta'}} \left[ \psi_\theta(x_1, u_1) \sum_{j=1}^{\infty} \beta^{j-1} r(x_j, u_j) \right],$$

*where $\beta \to 1^-$ denotes that $\beta$ converges to 1 from below.*

The proof is given in Section 4 in the supporting material. From the results in Lemmas 2 and 4, it is obvious that the estimator $\hat{\theta}_n$ given by Eq. (7) converges to the same solution as that of EMPS as $\beta \to 1$ from bellow. $\square$

Theorem 1 implies that WLPS monotonically increases the expected reward. It should be emphasized that WLPS provides us with an important insight into DPS, *i.e.*, the parameter update of EMPS can be interpreted as a well-studied maximum (weighted) likelihood estimation problem. This allows us to naturally apply various sophisticated techniques for model selection, which are well established in statistics, to DPS. In the next section, we discuss model selection for policy $\pi_\theta(u|x)$.

## 4 Model selection with WLPS

Common model selection strategies are carried out by comparing candidate models, which are specified in advance, based on a criterion that evaluates the goodness of fit of the model estimated from the obtained samples. Since the motivation for RL is to maximize the expected reward given in (2), it would be natural to seek an appropriate model for the policy through the computation of some reasonable measure to evaluate the expected reward from the sample sequences. However, since different policy models give different generative models for sample sequences, we need to obtain new sample sequences to evaluate the measure each time the model is changed. Therefore, employing a strategy of model selection based directly on the expected reward would be hopelessly inefficient.

Instead, to develop a tractable model selection, we focus on the weighted likelihood given by Eq. (5). As mentioned before, the policy with the maximum weighted log-likelihood must satisfy the maximum of the lower bound of the expected reward asymptotically. Moreover, since the weighted likelihood is defined under a certain fixed generative process for the sample sequences, unlike the expected reward case, the weighted likelihood can be evaluated using unique sample sequences even when the model has been changed. These observations lead to the fact that if it were possible to choose a good model from the candidate models in the sense of the weighted likelihood at each iteration in WLPS, we could realize an efficient DPS algorithm with model selection that achieves a monotonic increase in the expected reward.

In this study, we develop a criterion for choosing a suitable model by following the analogy of the *Bayesian information criterion* (BIC) [12], designed through asymptotic analysis of the posterior probability of the models given the data. Let $M_1, M_2, \cdots, M_k$ be $k$ candidate policy models, and assume that each model $M_j$ is characterized by a parametric policy $\pi_{\theta_j}(u|x)$ and the prior distribution $p(\theta_j|M_j)$ of the policy parameter. Also, define the *marginal weighted likelihood* of the $j$-th candidate model $\hat{p}_{\theta',j}(x_{2:n}, u_{1:n}|x_1)$ as

$$\hat{p}_{\theta',j}(x_{2:n}, u_{1:n}|x_1) := \int \pi_{\theta_j}(u_1|x_1)^{Q_1^\beta} \prod_{i=1}^n \pi_{\theta_j}(u_i|x_i)^{Q_i^\beta} p(x_i|x_{i-1}, u_{i-1}) p(\theta_j|M_j) \mathrm{d}\theta_j. \quad (10)$$

In a similar manner to the BIC, we now consider the posterior probability of the $j$-th model given the sample sequence by introducing the prior probability of the $j$-th model $p(M_j)$. From the generalized Bayes' rule, the posterior distribution of the $j$-th model is given by

$$p(M_j|x_{1:n}, u_{1:n}) := \frac{\hat{p}_{\theta',j}(x_{2:n}, u_{1:n}|x_1)p(M_j)}{\sum_{j'=1}^k \hat{p}_{\theta',j'}(x_{2:n}, u_{1:n}|x_1)p(M_{j'})}. \quad (11)$$

and in our model selection strategy, we adopt the model with the largest posterior probability.

For notational simplicity, in the following discussion we omit the subscript that represents the index indicating the number of models. Assuming that the prior probability is uniform in all models, the model with the maximum posterior probability corresponds to that of the marginal weighted likelihood. The behavior of the marginal weighted likelihood can be evaluated when the integrand of marginal weighted likelihood (10) is concentrated in a neighborhood of the weighted log-likelihood estimator given by estimating equation (7), as described in the following theorem.

**Theorem 2.** *Suppose that, given an initial state $x_1$, a random sequence $\{x_{2:n}, u_{1:n}\}$ is generated from the model $p_{\theta'}(x_{2:n}, u_{1:n}|x_1)$ of the MDP. Suppose that Assumptions 1-3 and 5 are satisfied. If the following conditions*

*(a) The estimator $\hat{\theta}_n$ given by Eq. (7) converges to $\theta$ at the rate of $O_p(n^{-1/2})$.*

*(b) The prior distribution $p(\theta|M)$ satisfies $p(\hat{\theta}_n|M) = O_p(1)$.*

*(c) The matrix $\mathrm{A}(\theta) := \mathbb{E}_{x_1 \sim \mu_{\theta'}}^{\pi_{\theta'}}[\mathrm{K}_\theta(x_1, u_1) \sum_{j=1}^\infty \beta^{j-i} r(x_j, u_j)]$ is invertible.*

*(d) For any $x \in \mathcal{X}$, $u \in \mathcal{U}$ and $\theta \in \Theta$, $K_\theta(x, u)$ is uniformly bounded.*

*are satisfied, the log marginal weighted likelihood can be calculated as*

$$\ln \hat{p}_{\theta'}(x_{2:n}, u_{1:n}|x_1) = L_n^{\theta'}(\hat{\theta}_n) - \frac{1}{2}m \ln n + O_p(1),$$

*where recall that $m$ denotes the number of dimensional of the model (policy parameter).*

The proof is given in Section 5 in the supporting material. Note that the term, $\sum_{i=2}^n \ln p(x_i|x_{i-1}, u_{i-1})$ in $L_n^{\theta'}(\hat{\theta}_n)$, does not depend on the model. Therefore, when evaluating the posterior probability of the model, it is sufficient to compute the following model selection criterion:

$$\mathcal{IC} = \sum_{i=1}^n \ln \pi_{\hat{\theta}_n}(u_i|x_i)^{Q_i^\beta} - \frac{1}{2}m \ln n. \quad (12)$$

As can be seen, this model selection criterion consists of two terms, where the first term is the weighted log-likelihood of the policy and the second is a penalty term that penalizes highly complex models. Also, since the first term is larger than the second term, this criterion gives the model with the maximum weighted log-likelihood asymptotically. Algorithm 2 describes the algorithm flow of WLPS including the model selection strategy.

**Algorithm 2** (WLPS with model selection)**.**

    *1. Generate a sample sequence $\{x_{1:n}, u_{1:n}\}$ by employing the current policy parameter $\theta$.*

    *2. For all models, find estimator $\hat{\theta}_n$ by solving estimating equation (7) and evaluate model selection criterion (12).*

    *3. Choose the best model based on model selection criterion (12) and check for convergence. If convergence is not satisfied, return to 1.*

**Empirical Example**   We evaluated the performance of the proposed model-selection method using a simple one-dimensional linear quadratic Gaussian (LQG) problem. This problem is known to be sufficiently difficult as an empirical evaluation, while it is analytically solvable. In this problem, we characterized the state transition distribution $p(x_i|x_{i-1}, u_{i-1})$ as a Gaussian distribution $N(x_i|\bar{x}_i, \sigma)$ with mean $\bar{x}_i = x_{i-1} + u_{i-1}$ and variance $\sigma = 0.5^2$. The reward function was set to a quadratic function $r(x_i, u_i) = -x_i^2 - u_i^2 + c$, where $c$ is a positive scalar value for preventing the reward $r(x, u)$ being negative. The control signal $u_i$ was generated from a Gaussian distribution $N(u_i|\bar{u}_i, \sigma')$ with mean $\bar{u}_i$ and variance $\sigma' = 0.5$. We used a linear model with polynomial basis functions defined as $\bar{u}_i = \sum_{j=1}^{k} \theta_j x_j^j + \theta_0$, where $k$ is the order of the polynomial. Note that, in this LQG setting, the optimal controller can be represented as a linear model, *i.e.*, the optimal policy can be obtained when the order of polynomial is selected as $k = 1$.

In this experiment, we validated whether the proposed model selection method can detect the true order of the polynomial. To illustrate how our proposed model selection criterion works, we compared the performance of the proposed model selection method with a naïve method based on the weighted log-likelihood (6). The weighted-log-likelihood-based selection, similarly to the proposed method, was performed by computing the weighted log-likelihood scores (6) over all candidate models and selecting the model with the maximum score among the candidates.

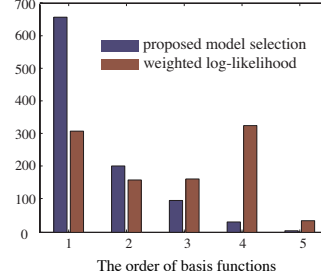

Figure 1: Distribution of order $k$ selected by our model selection criterion (left bar) and the weighted likelihood (right bar).

Figure 1 shows the distribution on the scores of the selected polynomial orders $k$ in the learned policies from first to fifth order by using the weighted log-likelihood and our model selection criterion. The distributions of the scores were obtained by repeating random 1000 trials. A learning process was performed by 200 iterations of WLPS, each of which contained 200 samples generated by the current policy. The discounted factor $\beta$ was set to 0.99. As shown in Figure 1, in the proposed method, the peak of the selected order was located at the true order $k = 1$. On the other hand, in the weighted log-likelihood method, the distribution of the orders converged to a one with two peaks at $k = 1$ and $k = 4$. This result seems to show that the penalized term in our model selection criterion worked well.

## 5   Discussion

In this study, we have discussed a DPS problem in the framework of weighted likelihood estimation. We introduced a weighted likelihood function as the objective function of DPS, and proposed an incremental algorithm, WLPS, based on the iteration of maximum weighted log-likelihood estimation. WLPS shows desirable theoretical properties, namely, consistency, asymptotic normality, and a monotonic increase in the expected reward at each iteration. Furthermore, we have constructed a model selection strategy based on the posterior probability of the model given a sample sequence through asymptotic analysis of the marginal weighted likelihood.

WLPS framework has a potential to bring a new theoretical insight to DPS, and derive more efficient algorithms based on the theoretical considerations. In the rest of this paper, we summarize some key issues that need to be addressed in future research.

### 5.1   Statistical interpretation of model-free and model-based WLPS

One of the important open issues in RL is how to combine model-free and model-based approaches with theoretical support. To this end, it is necessary to clarify the difference between model-based and model-free approaches in the theoretical sense. WLPS provides us with an interesting insight into the relation between model-free and model-based DPS from the viewpoint of statistics.

We begin by introducing the model-based WLPS method. Let us specify the state transition distribution $p(x'|x,u)$ as a parametric model $p_\kappa(x'|x,u) := p(x'|x,u,\kappa)$, where $\kappa$ is an $m'$-dimensional parameter vector. Assuming $p_\kappa(x'|x,u)$ with respect to parameter $\kappa$ and taking the partial derivative of the log weighted likelihood (6), we obtain the estimating equation for parameter $\kappa$:

$$\sum_{i=2}^{n} \xi_{\hat{\kappa}_n}(x_{i-1}, u_{i-1}, x_i) = 0, \tag{13}$$

where $\xi_\kappa(x, u, x')$ is the partial derivative of the state transition distribution $p_\kappa(x'|x,u)$ with respect to $\kappa$. As can be seen, estimating equation (13) corresponds to the likelihood equation, *i.e.*, the estimator, $\hat{\kappa}_n = \hat{\kappa}_n(x_{1:n}, u_{1:n-1})$, given by (13) is the maximum likelihood estimator. This observation indicates that the weighted likelihood integrates two different objective functions: one for learning policy $\pi_\theta(u|x)$, and the other for the state predictor, $p_\kappa(x'|x,u)$. Having obtained estimator $\hat{\kappa}_n$ from estimating equation (13), the model-based WLPS estimates the policy parameter by finding the solution, $\check{\theta}_n := \check{\theta}(x_{1:n}, u_{1:n})$, of the following estimating equation:

$$\iint p_{\theta', \hat{\kappa}_n}(x_{2:n}, u_{1:n}|x_1) \left\{ \sum_{i=1}^{n} \sum_{j=i}^{n} \beta^{j-i} \psi_{\check{\theta}_n}(x_i, u_i) r(x_j, u_j) \right\} \mathrm{d}x_{2:n} \mathrm{d}u_{1:n} = 0. \tag{14}$$

Note that estimating equation (14) is derived by taking the integral of Eq. (7) over the sample sequence $\{x_{2:n}, u_{1:n}\}$ based on the current estimated model $p_{\theta', \hat{\kappa}_n}(x_{2:n}, u_{1:n}|x_1)$. Thus, the model-based WLPS converges to the same parameter as the model-free WLPS, if model $p_\kappa(x'|x,u)$ is well specified[2].

We now consider the general treatment for model-free and model-based WLPS from a statistical viewpoint. Model-based WLPS fully specifies the weighted likelihood by using the parametric policy and parametric state transition models, and estimates all the parameters that appear in the parametric weighted likelihood. Hence, model-based WLPS can be framed as a parametric statistical inference problem. Meanwhile, model-free WLPS partially specifies the weighted likelihood by only using the parametric policy model. This can be seen as a *semiparametric statistical model* [22, 23], which includes not only parameters of interest, but also additional *nuisance parameters* with possibly infinite DoF, where only the policy is modeled parametrically and the other unspecified part corresponds to the nuisance parameters. Therefore, model-free WLPS can be framed as a *semiparametric statistical inference problem*. Hence, the difference between model-based and model-free WLPS methods can be interpreted as the difference between parametric and semiparametric statistical inference. The theoretical aspects of both parametric and semiparametric inference have been actively investigated and several approaches for combining their estimators have been proposed [23, 24, 25]. Therefore, by following these works, we have developed a novel hybrid DPS algorithm that combines model-free and model-based WLPS with desirable statistical properties.

### 5.2 Variance reduction technique for WLPS

In order to perform fast learning of the policy, it is necessary to find estimators that can reduce the estimation variance of the policy parameters in DPS. Although variance reduction techniques have been proposed in DPS [26, 27, 28], these employ indirect approaches, *i.e.*, instead of considering the estimation variance of the policy parameters, they reduce the estimation variance of the moments necessary to learn the policy parameter. Unfortunately, these variance reduction techniques do not guarantee decreasing the estimation variance of the policy parameters, and thus it is desirable to develop a direct approach that can evaluate and reduce the estimation variance of the policy parameters.

As stated above, we can interpret model-free WLPS as a semiparametric statistical inference problem. This interpretation allows us to apply the *estimating function method* [22, 23], which has been well established in semiparametric statistics, directly to WLPS. The estimating function method is a powerful tool for the design of consistent estimators and the evaluation of the estimation variance of parameters in a semiparametric inference problem. The advantage of considering the estimating function is the ability 1) to characterize an entire set of consistent estimators, and 2) to find the optimal estimator with the minimum parameter estimation variance from the set of estimators [23, 29]. Therefore, by applying this to WLPS, we can characterize an entire set of estimators, which maximizes the expected reward without identifying the state transition distribution, and find the optimal estimator with the minimum estimation variance.

## Footnotes

[1] In practice, the constant $\beta$ is set to an arbitrary value close to one. If we can analyze the finite sample behavior of the expected reward with the WLPS estimator, we may obtain a better estimator by finding an optimal $\beta$ in the sense of the maximization of the expected reward. Some researches have recently tackled to establish the finite sample analysis for RL based on statistical learning theory [20, 21]. These works might provide us with some insights into the finite sample analysis of WLPS.

[2]In the following discussion, in order to clarify the difference between the model-free and the model-based manners, we write original WLPS as model-free WLPS.

# References

[1] P. Dayan and G. Hinton, "Using expectation-maximization for reinforcement learning," *Neural Computation*, vol. 9, no. 2, pp. 271–278, 1997.

[2] J. Baxter and P. L. Bartlett, "Infinite-horizon policy-gradient estimation," *Journal of Artificial Intelligence Research*, vol. 15, no. 4, pp. 319–350, 2001.

[3] V. R. Konda and J. N. Tsitsiklis, "On actor-critic algorithms," *SIAM Journal on Control and Optimization*, vol. 42, no. 4, pp. 1143–1166, 2003.

[4] J. Peters and S. Schaal, "Reinforcement learning by reward-weighted regression for operational space control," in *Proceedings of the 24th International Conference on Machine Learning*, 2007.

[5] ——, "Natural actor-critic," *Neurocomputing*, vol. 71, no. 7-9, pp. 1180–1190, 2008.

[6] N. Vlassis, M. Toussaint, G. Kontes, and S. Piperidis, "Learning model-free robot control by a monte carlo em algorithm," *Autonomous Robots*, vol. 27, no. 2, pp. 123–130, 2009.

[7] E. Theodorou, J. Buchli, and S. Schaal, "A generalized path integral control approach to reinforcement learning," *Journal of Machine Learning Research*, vol. 11, pp. 3137–3181, 2010.

[8] J. Peters, K. Mülling, and Y. Altün, "Relative entropy policy search," in *Proceedings of the 24-th National Conference on Artificial Intelligence*, 2010.

[9] J. Kober and J. Peters, "Policy search for motor primitives in robotics," *Machine Learning*, vol. 84, no. 1-2, pp. 171–203, 2011.

[10] R. S. Sutton and A. G. Barto, *Reinforcement Learning: An Introduction*.   MIT Press, 1998.

[11] H. Akaike, "A new look at the statistical model identification," *IEEE Transactions on Automatic Control*, vol. 19, no. 6, pp. 716–723, 1974.

[12] G. Schwarz, "Estimating the dimension of a model," *The Annals of Statistics*, vol. 6, no. 2, pp. 461–464, 1978.

[13] A. Farahmand and C. Szepesvári, "Model selection in reinforcement learning," *Machine Learning*, pp. 1–34, 2011.

[14] M. M. Fard and J. Pineau, "PAC-Bayesian model selection for reinforcement learning," in *Advances in Neural Information Processing Systems 22*, 2010.

[15] H. Hachiya, J. Peters, and M. Sugiyama, "Reward-weighted regression with sample reuse for direct policy search in reinforcement learning," *Neural Computation*, vol. 23, no. 11, pp. 2798–2832, 2011.

[16] M. G. Azar and H. J. Kappen, "Dynamic policy programming," Tech. Rep. arXiv:1004.202, 2010.

[17] H. Kappen, V. Gómez, and M. Opper, "Optimal control as a graphical model inference problem," *Machine learning*, pp. 1–24, 2012.

[18] K. Rawlik, M. Toussaint, and S. Vijayakumar, "On stochastic optimal control and reinforcement learning by approximate inference," in *International Conference on Robotics Science and Systems*, 2012.

[19] R. C. Bradley, "Basic properties of strong mixing conditions. A survey and some open questions," *Probability Surveys*, vol. 2, pp. 107–144, 2005.

[20] R. Munos and C. Szepesvári, "Finite-time bounds for fitted value iteration," *Journal of Machine Learning Research*, vol. 9, pp. 815–857, 2008.

[21] A. Lazaric, M. Ghavamzadeh, and R. Munos, "Finite-sample analysis of least-squares policy iteration," *Journal of Machine Learning Research*, vol. 13, p. 30413074, 2012.

[22] V. P. Godambe, Ed., *Estimating Functions*.   Oxford University Press, 1991.

[23] S. Amari and M. Kawanabe, "Information geometry of estimating functions in semi-parametric statistical models," *Bernoulli*, vol. 3, no. 1, pp. 29–54, 1997.

[24] P. J. Bickel, C. A. Klaassen, Y. Ritov, and J. A. Wellner, *Efficient and Adaptive Estimation for Semiparametric Models*.   Springer, 1998.

[25] G. Bouchard and B. Triggs, "The tradeoff between generative and discriminative classifiers," in *Proceedings 1998 16th IASC International Symposium on Computational Statistics*, 2004, pp. 721–728.

[26] E. Greensmith, P. L. Bartlett, and J. Baxter, "Variance reduction techniques for gradient estimates in reinforcement learning," *Journal of Machine Learning Research*, vol. 5, pp. 1471–1530, 2004.

[27] R. Munos, "Geometric variance reduction in markov chains: application to value function and gradient estimation," *Journal of Machine Learning Research*, vol. 7, pp. 413–427, 2006.

[28] T. Zhao, H. Hachiya, G. Niu, and M. Sugiyama, "Analysis and improvement of policy gradient estimation," *Neural Networks*, 2011.

[29] T. Ueno, S. Maeda, M. Kawanabe, and S. Ishii, "Generalized TD learning," *Journal of Machine Learning Research*, vol. 12, pp. 1977–2020, 2011.

